# A Topographic Product for the Optimization of Self-Organizing Feature Maps

**Hans-Ulrich Bauer, Klaus Pawelzik, Theo Geisel**
Institut für theoretische Physik and SFB Nichtlineare Dynamik
Universität Frankfurt
Robert-Mayer-Str. 8-10
W-6000 Frankfurt 11
Fed. Rep. of Germany
email: bauer@asgard.physik.uni-frankfurt.dbp

## Abstract

Optimizing the performance of self-organizing feature maps like the Kohonen map involves the choice of the output space topology. We present a topographic product which measures the preservation of neighborhood relations as a criterion to optimize the output space topology of the map with regard to the global dimensionality $D^A$ as well as to the dimensions in the individual directions. We test the topographic product method not only on synthetic mapping examples, but also on speech data. In the latter application our method suggests an output space dimensionality of $D^A = 3$, in coincidence with recent recognition results on the same data set.

## 1 INTRODUCTION

Self-organizing feature maps like the Kohonen map (Kohonen, 1989, Ritter et al., 1990) not only provide a plausible explanation for the formation of maps in brains, e.g. in the visual system (Obermayer et al., 1990), but have also been applied to problems like vector quantization, or robot arm control (Martinetz et al., 1990). The underlying organizing principle is the preservation of neighborhood relations. For this principle to lead to a most useful map, the topological structure of the output space must roughly fit the structure of the input data. However, in technical

applications this structure is often not a priory known. For this reason several attempts have been made to modify the Kohonen-algorithm such, that not only the weights, but also the output space topology itself is adapted during learning (Kangas et al., 1990, Martinetz et al., 1991).

Our contribution is also concerned with optimal output space topologies, but we follow a different approach, which avoids a possibly complicated structure of the output space. First we describe a quantitative measure for the preservation of neighborhood relations in maps, the topographic product $P$. The topographic product had been invented under the name of "wavering product" in nonlinear dynamics in order to optimize the embeddings of chaotic attractors (Liebert et al., 1991). $P = 0$ indicates perfect match of the topologies. $P < 0$ ($P > 0$) indicates a folding of the output space into the input space (or vice versa), which can be caused by a too small (resp. too large) output space dimensionality. The topographic product can be computed for any self-organizing feature map, without regard to its specific learning rule. Since judging the degree of twisting and folding by visually inspecting a plot of the map is the only other way of "measuring" the preservation of neighborhoods, the topographic product is particularly helpful, if the input space dimensionality of the map exceeds $D^A = 3$ and the map can no more be visualized. Therefore the derivation of the topographic product is already of value by itself.

In the second part of the paper we demonstrate the use of the topographic product by two examples. The first example deals with maps from a 2D input space with nonflat stimulus distribution onto rectangles of different aspect ratios, the second example with the map of 19D speech data onto output spaces of different dimensionality. In both cases we show, how the output space topology can be optimized using our method.

# 2    DERIVATION OF THE TOPOGRAPHIC PRODUCT

## 2.1    KOHONEN-ALGORITHM

In order to introduce the notation necessary to derive the topographic product, we very briefly recall the Kohonen algorithm. It describes a map from an input space $\mathbf{V}$ into an output space $A$. Each node $j$ in $A$ has a weight vector $\mathbf{w}_j$ associated with it, which points into $\mathbf{V}$. A stimulus $\mathbf{v}$ is mapped onto that node $i$ in the output space, which minimizes the input space distance $d^V(\mathbf{w}_i, \mathbf{v})$:

$$i: \quad d^V(\mathbf{w}_i, \mathbf{v}) = \min_{j \in A} d^V(\mathbf{w}_j, \mathbf{v}). \tag{1}$$

During a learning step, a random stimulus is chosen in the input space and mapped onto an output node $i$ according to Eq. 1. Then all weights $\mathbf{w}_j$ are shifted towards $\mathbf{v}$, with the amount of shift for each weight vector being determined by a neighborhood function $h_{i,j}^0$:

$$\delta\mathbf{w}_j = \epsilon h_{j,i}^0(d^A(j,i))(\mathbf{v} - \mathbf{w}_j) \quad \forall j \in A. \tag{2}$$

($d^A(j,i)$ measures distances in the output space.) $h_{j,i}^0$ effectively restricts the nodes participating in the learning step to nodes in the vicinity of $i$. A typical choice for

the neighborhood function is

$$h_{j,i}^0 = \exp\left(-\frac{(d^A)^2(j,i)}{2\sigma^2}\right). \tag{3}$$

In this way the neighborhood relations in the output space are enforced in the input space, and the output space topology becomes of crucial importance. Finally it should be mentioned that the learning step size $\epsilon$ as well as the width of the neighborhood function $\sigma$ are decreased during the learning for the algorithm to converge to an equilibrium state. A typical choice is an exponential decrease. For a detailed discussion of the convergence properties of the algorithm, see (Ritter et al., 1988).

## 2.2  TOPOGRAPHIC PRODUCT

After the learning phase, the topographic product is computed as follows. For each output space node $j$, the nearest neighbor ordering in input space and output space is computed ($n_k^A(j)$ denotes the $k$-th nearest neighbor of $j$ in $A$, $n_k^V(j)$ in $V$). Using these quantities, we define the ratios

$$Q_1(j,k) = \frac{d^V(\mathbf{w}_j, \mathbf{w}_{n_k^A(j)})}{d^V(\mathbf{w}_j, \mathbf{w}_{n_k^V(j)})}, \tag{4}$$

$$Q_2(j,k) = \frac{d^A(j, n_k^A(j))}{d^A(j, n_k^V(j))}. \tag{5}$$

One has $Q_1(j,k) = Q_2(j,k) = 1$ only, if the $k$-th nearest neighbors in $V$ and $A$ coincide. Any deviations of the nearest neighbor ordering will result in values for $Q_{1,2}$ deviating from 1. However, not all differences in the nearest neighbor orderings in $V$ and $A$ are necessarily induced by neighborhood violations. Some can be due to locally varying magnification factors of the map, which in turn are induced by spatially varying stimulus densities in $V$. To cancel out the latter effects, we define the products

$$P_1(j,k) = \left(\Pi_{l=1}^k Q_1(j,l)\right)^{\frac{1}{k}}, \tag{6}$$

$$P_2(j,k) = \left(\Pi_{l=1}^k Q_2(j,l)\right)^{\frac{1}{k}}. \tag{7}$$

For these the relations

$$P_1(j,k) \geq 1,$$
$$P_2(j,k) \leq 1$$

hold. Large deviations of $P_1$ (resp. $P_2$) from the value 1 indicate neighborhood violations, when looking from the output space into the input space (resp. from the input space into the output space). In order to get a symmetric overall measure, we further multiply $P_1$ and $P_2$ and find

$$P_3(j,k) = \left(\Pi_{l=1}^k Q_1(j,k)Q_2(j,k)\right)^{\frac{1}{2k}}. \tag{8}$$

Further averaging over all nodes and neighborhood orders finally yields the topographic product

$$P = \frac{1}{N(N-1)} \sum_{j=1}^{N} \sum_{k=1}^{N-1} \log(P_3(j,k)). \tag{9}$$

The possible values for $P$ are to be interpreted as follows:

$$
\begin{aligned}
P &\leq 0: & \text{output space dimension } D^A \text{ too low,} \\
P &= 0: & \text{output space dimension } D^A \text{ o.k.,} \\
P &\geq 0: & \text{output space dimension } D^A \text{ too high.}
\end{aligned}
$$

These formulas suffice to understand how the product is to be computed. A more detailed explanation for the rational behind each individual step of the derivation can be found in a forthcoming publication (Bauer et al., 1991).

## 3    EXAMPLES

We conclude the paper with two examples which exemplify how the method works.

### 3.1    ILLUSTRATIVE EXAMPLE

The first example deals with the mapping from a 2D input space onto rectangles of different aspect ratios. The stimulus distribution is flat in one direction, Gaussian shaped in the other (Fig 1a). The example demonstrates two aspects of our method at once. First it shows that the method works fine with maps resulting from nonflat stimulus distributions. These induce spatially varying areal magnification factors of the map, which in turn lead to twists in the neighborhood ordering between input space and output space. Compensation for such twists was the purpose of the multiplication in Eqs (6) and (7).

Table 1: Topographic product $P$ for the map from a square input space with a Gaussian stimulus distribution in one direction, onto rectangles with different aspect ratios. The values for $P$ are averaged over 8 networks each. The 43×6-output space matches the input data best, since its topographic product is smallest.

| $N$ | aspect ratio | $P$ |
|---|---|---|
| 256×1 | 256 | -0.04400 |
| 128×2 | 64 | -0.03099 |
| 64×4 | 16 | -0.00721 |
| 43×6 | 7.17 | 0.00127 |
| 32×8 | 4 | 0.00224 |
| 21×12 | 1.75 | 0.01335 |
| 16×16 | 1 | 0.02666 |

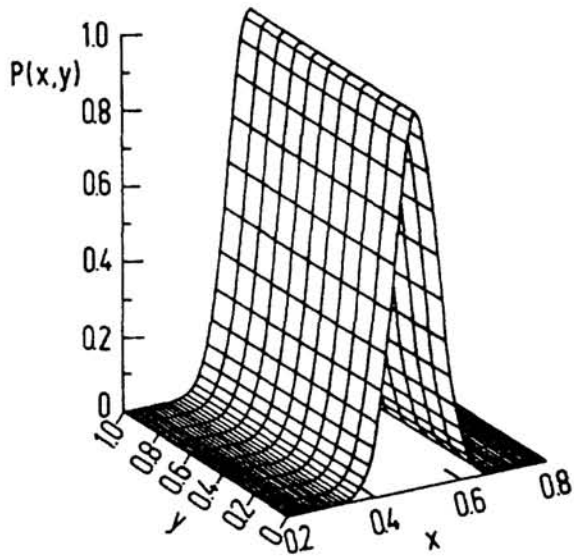

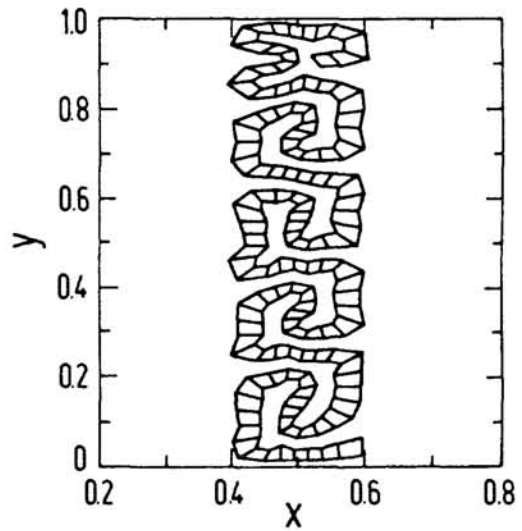

Fig. 1a                                                                    Fig. 1b

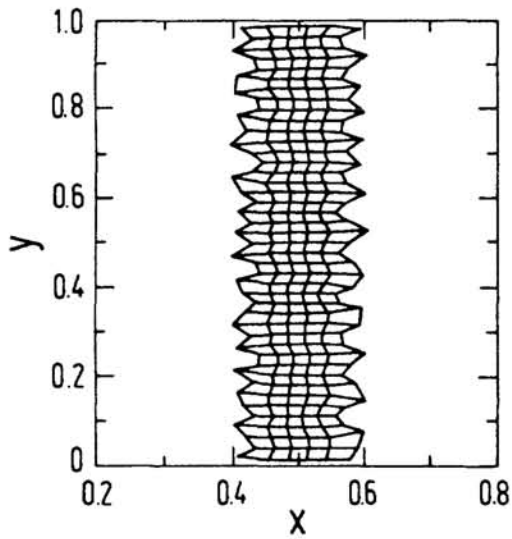

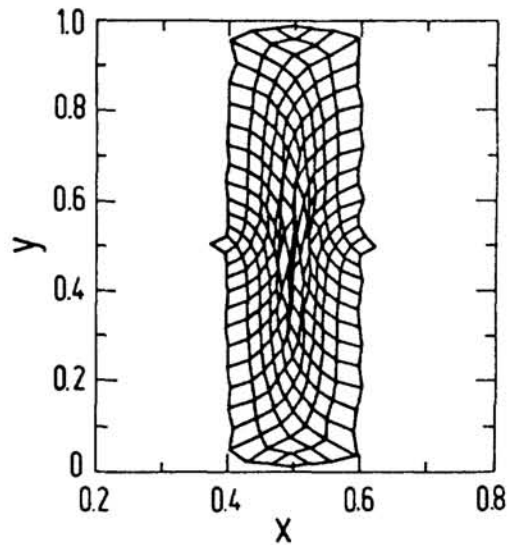

Fig. 1c                                                                    Fig. 1d

Figure 1: Self-organizing feature maps of a Gaussian shaped (a) 2-dimensional stimulus distribution onto output spaces with $128 \times 2$ (b), $43 \times 6$ (c) and $16 \times 16$ (d) output nodes. The $43 \times 6$-output space preserves neighborhood relations best.

Secondly the method cannot only be used to optimize the overall output space dimensionality, but also the individual dimensions in the different directions (i.e. the different aspect ratios). If the rectangles are too long, the resulting map is folded like a Peano curve (Fig. 1b), and neighborhood relations are severely violated perpendicular to the long side of the rectangle. If the aspect ratio fits, the map has a regular look (Fig. 1c), neighborhoods are preserved. The zig-zag-form at the outer boundary of the rectangle does not correspond to neighborhood violations. If the rectangle approaches a square, the output space is somewhat squashed into the input space, again violating neighborhood relations (Fig. 1d). The topographic product $P$ coincides with this intuitive evaluation (Tab. 1) and picks the $43 \times 6$-net as the most neighborhood preserving one.

## 3.2   APPLICATION EXAMPLE

In our second example speech data is mapped onto output spaces of various dimensionality. The data represent utterances of the ten german digits, given as 19-dimensional acoustical feature vectors (Gramß et al., 1990). The $P$-values for the different maps are given in Tab. 2. For both the speaker-dependent as well as the speaker-independent case the method distinguishes the maps with $D^A = 3$ as most neighborhood preserving. Several points are interesting about these results. First of all, the suggested output space dimensionality exceeds the widely used $D^A = 2$. Secondly, the method does not generally judge larger output space dimensions as more neighborhood preserving, but puts an upper bound on $D^A$. The data seems to occupy a submanifold of the input space which is distinctly lower than four dimensional. Furthermore we see that the transition from one to several speakers does not change the value of $D^A$ which is optimal under neighborhood considerations. This contradicts the expectation that the additional interspeaker variance in the data occupies a full additional dimension.

Table 2: Topographic product $P$ for maps from speech feature vectors in a 19D input space onto output spaces of different dimensionality $D^{\mathbf{V}}$.

| $D^{\mathbf{V}}$ | $N$ | $P$ speaker-dependent | $P$ speaker-independent |
|---|---|---|---|
| 1 | 256 | -0.156 | -0.229 |
| 2 | 16×16 | -0.028 | -0.036 |
| 3 | 7×6×6 | 0.019 | 0.007 |
| 4 | 4×4×4×4 | 0.037 | 0.034 |

What do these results mean for speech recognition? Let us suppose that several utterances of the same word lead to closeby feature vector sequences in the input space. If the mapping was not neighborhood preserving, one should expect the trajectories in the output space to be separated considerably. If a speech recognition system compares these output space trajectories with reference trajectories corresponding to reference utterances of the words, the probability of misclassification rises. So one should expect that a word recognition system with a Kohonen-map

preprocessor and a subsequent trajectory classifier should perform better if the neighborhoods in the map are preserved.

The results of a recent speech recognition experiment coincide with these heuristic expectations (Brandt et al., 1991). The experiment was based on the same data set, made use of a Kohonen feature map as a preprocessor, and of a dynamic time-warping algorithm as a sequence classifier. The recognition performance of this hybrid system turned out to be better by about 7% for a 3D map, compared to a 2D map with a comparable number of nodes (0.795 vs. 0.725 recognition rate).

## Acknowledgements

This work was supported by the Deutsche Forschungsgemeinschaft through SFB 185 "Nichtlineare Dynamik", TP A10.

## References

H.-U. Bauer, K. Pawelzik, Quantifying the Neighborhood Preservation of Self-Organizing Feature Maps, submitted to IEEE TNN (1991).

W.D. Brandt, H. Behme, H.W. Strube, Bildung von Merkmalen zur Spracherkennung mittels Phonotopischer Karten, Fortschritte der Akustik - Proc. of DAGA 91 (DPG GmbH, Bad Honnef), 1057 (1991).

T. Gramß, H.W. Strube, Recognition of Isolated Words Based on Psychoacoustics and Neurobiology, Speech Comm. **9**, 35 (1990).

J.A. Kangas, T.K. Kohonen, J.T. Laaksonen, Variants of Self-Organizing Maps, IEEE Trans. Neur. Net. **1**, 93 (1990).

T. Kohonen, Self-Organization and Associative Memory, 3rd Ed., Springer (1989).

W. Liebert, K. Pawelzik, H.G. Schuster, Optimal Embeddings of Chaotic Attractors from Topological Considerations, Europhysics Lett. **14**, 521 (1991).

T. Martinetz, H. Ritter, K. Schulten, Three-Dimensional Neural Net for Learning Visuomotor Coordination of a Robot Arm, IEEE Trans. Neur. Net. **1**, 131 (1990).

T. Martinetz, K. Schulten, A "Neural-Gas" Network Learns Topologies, Proc. ICANN 91 Helsinki, ed. Kohonen et al., North-Holland, I-397 (1991).

K. Obermaier, H. Ritter, K. Schulten, A Principle for the Formation of the Spatial Structure of Cortical Feature Maps, Proc. Nat. Acad. Sci. USA **87**, 8345 (1990).

H. Ritter, K. Schulten, Convergence Properties of Kohonen's Topology Conserving Maps: Fluctuations, Stability and Dimension Selection, Biol. Cyb. **60**, 59-71 (1988).

H. Ritter, T. Martinetz, K. Schulten, Neuronale Netze, Addison Wesley (1990).
